# Assessing and Improving Neural Network Predictions by the Bootstrap Algorithm

**Gerhard Paass**
German National Research Center for Computer Science (GMD)
D-5205 Sankt Augustin, Germany
e-mail: **paass@gmd.de**

## Abstract

The bootstrap algorithm is a computational intensive procedure to derive nonparametric confidence intervals of statistical estimators in situations where an analytic solution is intractable. It is applied to neural networks to estimate the predictive distribution for unseen inputs. The consistency of different bootstrap procedures and their convergence speed is discussed. A small scale simulation experiment shows the applicability of the bootstrap to practical problems and its potential use.

## 1   INTRODUCTION

Bootstrapping is a strategy for estimating standard errors and confidence intervals for parameters when the form of the underlying distribution is unknown. It is particularly valuable when the parameter of interest is a complicated functional of the true distribution. The key idea first promoted by Efron (1979) is that the relationship between the true cumulative distribution function (cdf) $F$ and the sample of size $n$ is similar to the relationship between the empirical cdf $\hat{F}_n$ and a secondary sample drawn from it. So one uses the primary sample to form an estimate $\hat{F}_n$ and calculates the sampling distribution of the parameter estimate under $\hat{F}_n$. This calculation is done by drawing many secondary samples and finding the estimate, or function of the estimate, for each. If $\mathcal{F}_n$ is a good approximation of $F$, then $H_n$, the sampling distribution of the estimate under $\hat{F}_n$, is a generally good approximation to the sampling distribution for the estimate under $F$. $H_n$ is

called the *bootstrap distribution* of the parameter. Introductory articles are Efron and Gong (1983) and Efron and Tibshirani (1986). For a survey of bootstrap results see Hinkley (1988) and DiCiccio and Romano (1988).

A neural networks often may be considered as a nonlinear or nonparametric regression model

$$z = g_\beta(y) + \epsilon \tag{1}$$

which defines the relation between the vectors $y$ and $z$ of input and output variables. The term $\epsilon$ can be interpreted as a random 'error' and the function $g_\beta$ depends on some unknown parameter $\beta$ which may have infinite dimension. Usually the network is used to determine a prediction $z_0 = g_\beta(y_0)$ for some new input vector $y_0$. If the data is a random sample, an estimate $\hat{\beta}$ differs from the true value of $\beta$ because of the sampling error and consequently the prediction $g_{\hat{\beta}}(y_0)$ is different from the true prediction. In this paper the bootstrap approach is used to approximate a sampling distribution of the prediction (or a function thereof) and to estimate parameters of that distribution like its mean value, variance, percentiles, etc. Bootstrapping procedures are closely related to other resampling methods like cross validation and jackknife (Efron 1982). The jackknife can be considered as a linear approximation to the bootstrap (Efron, Tibshirani 1986).

In the next section different versions of the bootstrap procedure for feedforward neural networks are defined and their theoretical properties are reviewed. Main points are the convergence of the bootstrap distribution to true theoretical distribution and the speed of that convergence. In the following section the results of a simulation experiment for a simple backprop model are reported and the application of the bootstrap to model selection is discussed. The final section gives a short summary.

## 2  CONSISTENCY OF THE BOOTSTRAP FOR FEEDFORWARD NEURAL NETWORKS

Assume $X(n) := (x_1, \ldots, x_n)$ is the available independent, identically distributed (iid) sample from an underlying cdf $F$ where $x_i = (z_i, y_i)$ and $\hat{F}_n$ is the corresponding empirical cdf. For a given $y_0$ let $\eta = \eta(g_\beta(y_0))$ be a parameter of interest of the prediction, e.g. the mean value of the prediction of a component of $z$ for $y_0$.

The *pairwise bootstrap* algorithm is an intuitive way to apply the bootstrap notion to regression. It was proposed by Efron (1982) and involves the independent repetition of following steps for $b = 1, \ldots, B$.

  1. A sample $X_b^*(n)$ of size $n$ is generated from $\hat{F}_n$. Notice that this amounts to the random selection of $n$ elements from $X(n)$ *with replacement*.

  2. An estimate $\hat{\eta}_b$ is determined from $X_b^*(n)$.

The resulting empirical cdf of the $\hat{\eta}_b$, $b = 1, \ldots, n$ is denoted by $\hat{H}_B$ and approximates the sampling distribution for the estimate $\hat{\eta}$ under $\hat{F}_n$. The standard deviation of $H_B$ is an estimate of the standard error of $\eta(\hat{F}_n)$, and $[\hat{H}_B^{-1}(\alpha), \hat{H}_B^{-1}(1-\alpha)]$ is an approximate $(1 - 2\alpha)$ central confidence interval.

In general two conditions are necessary for the bootstrap to be consistent:

- The estimator, e.g. $\hat{\eta}_b$ has to be consistent.
- The functional which maps $F$ to $\hat{H}_B$ has to be smooth.

This requirement can be formalized by a uniform weak convergence condition (Di-Ciccio, Romano 1988). Using these concepts Freedman (1981) proved that for the parameters of a linear regression model the pairwise bootstrap procedure is consistent, i.e. yields the desired limit distribution for $n, B \to \infty$. Mammen (1991) showed that this also holds for the predictive distribution of a linear model (i.e. linear contrasts). These results hold even if the errors are heteroscedastic, i.e. if the distribution of $\epsilon_i$ depends on the value of $y_i$.

The performance of the bootstrap for *linear* regression is extensively discussed by Wu (1986). It turns out that the small sample properties can be different from the asymptotic relations and the bootstrap may exhibit a sizeable bias. Various procedures of bias correction have been proposed (DiCiccio, Romano 1988). Beran (1990) discusses a calibrated bootstrap prediction region containing the prediction $g_\beta(y_0)+\epsilon$ with prescribed probability $\alpha$. It requires a sequence of nested bootstraps. Its coverage probability tends to $\alpha$ at a rate up to $n^{-2}$. Note that this procedure can be applied to *nonlinear* regression models (1) with homoscedastic errors (Example 3 in Beran (1990, p.718) can be extended to this case).

Biases especially arise if the errors are heteroscedastic. Hinkley (1988) discusses the parametric modelling of dependency of the error distribution (or its variance) from $y$ and the application of the bootstrap algorithm using this model. The problem is here to determine this parametric dependency from the data. As an alternative Wu (1986) and Liu (1988) take into account heteroscedasticity in a nonparametric way. They propose the following *wild bootstrap* algorithm which starts with a consistent estimate $\hat{\beta}$ based on the sample $X(n)$. Then the set of residuals $(\hat{\epsilon}_1, \dots, \hat{\epsilon}_n)$ with $\hat{\epsilon}_i := z_i - g_{\hat{\beta}}(y_i)$ is determined. The approach attempts to mimic the conditional distribution of $z$ given $y_i$ in a very crude way by defining a distribution $\hat{G}_i$ whose first three moments coincide with the observed residual $\hat{\epsilon}_i$:

$$\int u d\hat{G}_i(u) = 0 \qquad \int u^2 d\hat{G}_i(u) = \hat{\epsilon}_i^2 \qquad \int u^3 d\hat{G}_i(u) = \hat{\epsilon}_i^3 \qquad (2)$$

Two point distributions are used which are uniquely defined by this requirement (Mammen 1991, p.121). Then the following steps are repeated for $b = 1, \dots, B$:

1. Independently generate residuals $\tilde{\epsilon}_i$ according to $\hat{G}_i$ and generate observations $z_i^* := g_{\hat{\beta}}(y_i) + \tilde{\epsilon}_i$ for $i = (1, \dots, n)$. This yields a new sample $X_b^*(n)$ of size $n$.

2. An estimate $\hat{\eta}_b^*$ is determined from $X_b^*(n)$.

The resulting empirical cdf of the $\hat{\eta}_b^*$ is then taken as the bootstrap distribution $\hat{H}_B$ which approximates the sampling distribution for the estimate $\hat{\eta}$ under $\hat{F}_n$. Mammen (1991, p.123) shows that this algorithm is consistent for the prediction of linear regression models if the least square estimator or M-estimators are used and discusses the convergence speed of the procedure.

The bootstrap may also be applied to nonparameric regression models like kernel-type estimators of the form

$$\hat{g}(y) = \frac{\left[\sum_{i=1}^{n} z_i K\left(\frac{y-y_i}{h}\right)\right]}{\left[\sum_{i=1}^{n} K\left(\frac{y-y_i}{h}\right)\right]} \tag{3}$$

with kernel $K$ and bandwidth $h$. These models are related to radial basis functions discussed in the neural network literature. For those models the pairwise bootstrap does not work (Härdle, Mammen 1990) as the algorithm is not forced to perform local a. əraging. To account for heteroscedasticity in the errors of (1) Härdle (1990, p.103) advocates the use of the wild bootstrap algorithm described above. Under some regularity conditions he shows the convergence of the bootstrap distribution of the kernel estimator to the correct limit distribution.

To summarize the bootstrap often is used simply because an analytic derivation of the desired sampling distribution is too complicated. The asymptotic investigations offer two additional reasons:

- There exist versions of the bootstrap algorithm that have a better rate of convergence than the usual asymptotic normal approximation. This effect has been extensively discussed in literature e.g. by Hall (1988), Beran (1988), DiCiccio and Romano (1988, p.349), Mammen (1991, p.74).

- There are cases where the bootstrap works, even if the normal approximation breaks down. Bickel and Freedman (1983) for instance show, that the bootstrap is valid for linear regression models in the presence of outliers and if the number of parameters changes with $n$. Their results are discussed and extended by Mammen (1991, p.88ff).

## 3   SIMULATION EXPERIMENTS

To demonstrate the performance of the bootstrap for real real problems we investigated a small neural network. To get a nonlinear situation we chose a "noisy" version of the xor model with eight input units $y_1, \ldots, y_8$ and a single output unit $z$. The input variables may take the values 0 and 1. The output unit of the true model is stochastic. It takes the values 0.1 and 0.9 with the following probabilities:

$$
\begin{aligned}
p(y=0.9) &= 0.9 \quad \text{if} \quad x_1 + x_2 + x_3 + x_4 < 3 \quad \text{and} \quad x_5 + x_6 + x_7 + x_8 < 3 \\
p(y=0.9) &= 0.1 \quad \text{if} \quad x_1 + x_2 + x_3 + x_4 < 3 \quad \text{and} \quad x_5 + x_6 + x_7 + x_8 \geq 3 \\
p(y=0.9) &= 0.1 \quad \text{if} \quad x_1 + x_2 + x_3 + x_4 \geq 3 \quad \text{and} \quad x_5 + x_6 + x_7 + x_8 < 3 \\
p(y=0.9) &= 0.9 \quad \text{if} \quad x_1 + x_2 + x_3 + x_4 \geq 3 \quad \text{and} \quad x_5 + x_6 + x_7 + x_8 \geq 3
\end{aligned}
$$

In contrast to the simple xor model generalization is possible in this setup. We generated a training set $X(n)$ of $n = 100$ inputs using the true model.

We used the pairwise bootstrap procedure described above and generated $B = 30$ different bootstrap samples $X_b^*(n)$ by random selection from $X(n)$ with replacement. This number of bootstrap samples is rather low and only will yield reliable information on the central tendency of the prediction. More sensitive parameters of

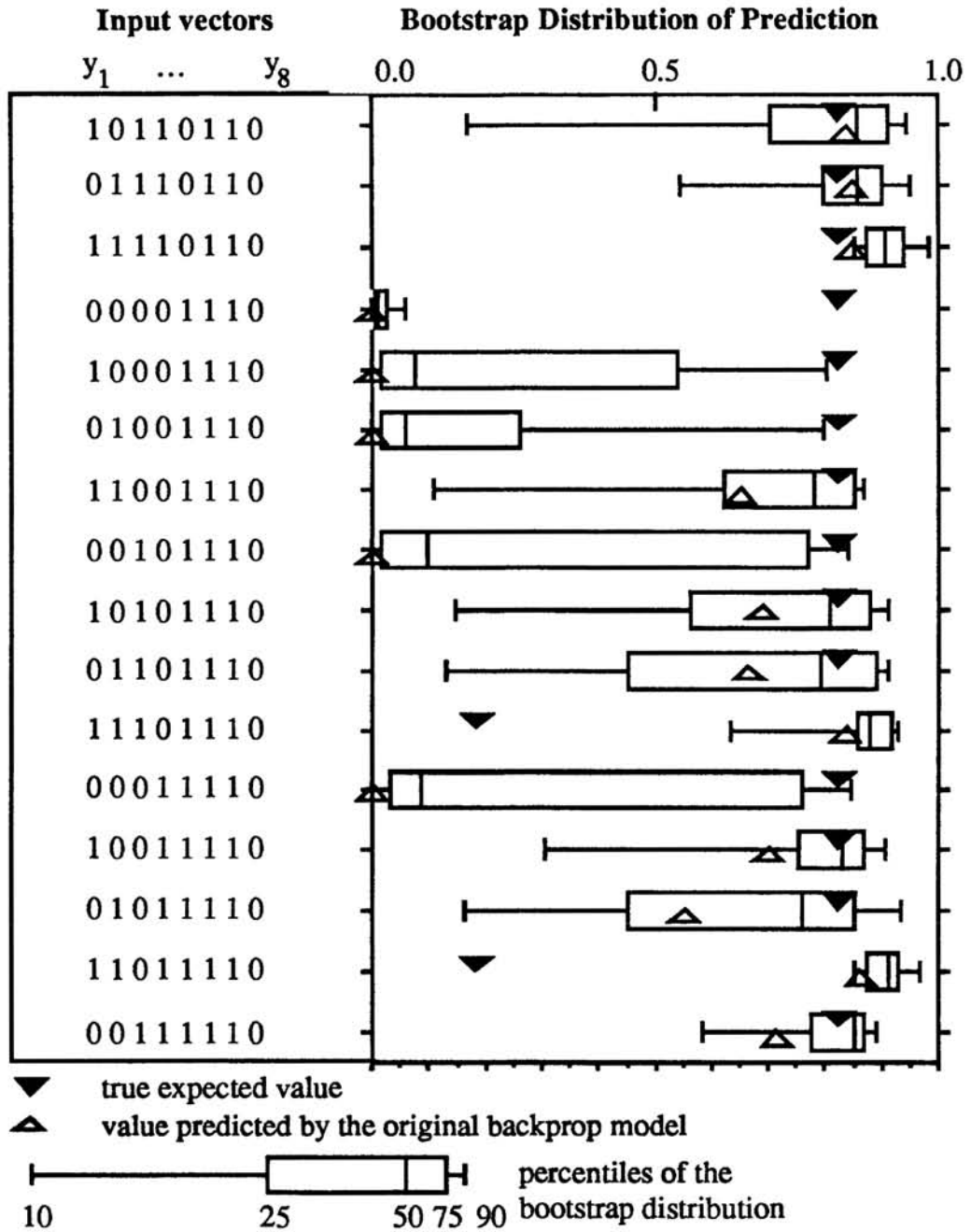

Figure 1: **Box-Plots of the Bootstrap Predictive Distribution for a Series of Different Input Vectors**

the distribution like low percentiles and the standard deviation can be expected to exhibit larger fluctuations. We estimated 30 weight vectors $\hat{\beta}_b$ from those samples by the backpropagation method with random initial weights. Subsequently for each of the 256 possible input vectors $y_i$ we determined the prediction $g_{\hat{\beta}_b}(y_i)$ yielding a predictive distribution. For comparison purposes we also estimated the weights of the original backprop model with the full data set $X(n)$ and the corresponding

Table 1: Mean Square Deviation from the True Prediction

| INPUT TYPE | HIDDEN UNITS | MEAN SQUARE DIFFERENCE | |
|---|---|---|---|
| | | BOOTSTRAP $D_B$ | FULL DATA $D_F$ |
| training inputs | 2 | 0.18 | 0.19 |
| | 3 | 0.17 | 0.19 |
| | 4 | 0.17 | 0.19 |
| non-training inputs | 2 | 0.30 | 0.34 |
| | 3 | 0.35 | 0.38 |
| | 4 | 0.37 | 0.42 |

Table 2: Coverage Probabilities of the Bootstrap Confidence Interval for Prediction

| HIDDEN UNITS | FRACTION OF CASES WITH TRUE PREDICTION IN | |
|---|---|---|
| | $[q_{25}, q_{75}]$ | $[q_{10}, q_{90}]$ |
| 2 | 0.47 | 0.77 |
| 3 | 0.44 | 0.70 |
| 4 | 0.43 | 0.70 |

predictions.

For some of those input vectors the results are shown in figure 1. The distributions differ greatly in size and form for the different input vectors. Usually the spread of the predictive distribution is large if the median prediction differs substantially from the true value. This reflects the situation that the observed data does not have much information on the specific input vector. Simply by inspecting the predictive distribution the reliability of a predictions may be assessed in a heuristic way. This may be a great help in practical applications.

In table 1 the mean square difference $D_B := \left(\frac{1}{n}\sum_{i=1}^{n}(z_i - q_{50})^2\right)^{1/2}$ between the true prediction $z_i$ and the median $q_{50}$ of the bootstrap predictive distribution is compared to the mean square difference $D_S := \left(\frac{1}{n}\sum_{i=1}^{n}(z_i - \hat{z}_{i,F})^2\right)^{1/2}$ between the true prediction and the value $\hat{z}_{i,F}$ estimated with full data backprop model. For the non-training inputs the bootstrap median has a lower mean deviation from the true value. This effect is a real practical advantage and occurs even for this simple bootstrap procedure. It may be caused in part by the variation of the initial weight values (cf. Pearlmutter, Rosenfeld 1991). The utilization of bootstrap procedures with higher order convergence has the potential to improve this effect.

Table 2 list the fraction of cases in the full set of all 256 possible inputs where the true value is contained in the central 50% and 80% prediction interval. Note that the intervals are based on only 30 cases. For the correct model with 2 hidden units the difference is 0.03 which corresponds to just one case. Models with more hidden units exhibit larger fluctuations. To arrive at more reliable intervals the number of

Table 3: Spread of the Predictive Distribution

| HIDDEN UNITS | MEAN INTERQUARTILE RANGE FOR ||
| | TRAINING INPUTS | NON-TRAINING INPUTS |
| --- | --- | --- |
| 2 | 0.13 | 0.29 |
| 3 | 0.11 | 0.35 |
| 4 | 0.11 | 0.37 |

bootstrap samples has to be increased by an order of magnitude.

If we use a model with more than two hidden units the fit to the training sample cannot be improved but remains constant. For nontraining inputs, however, the predictions of the model deteriorate. In table 1 we see that the mean square deviation from the true prediction increases. This is just a manifestation of 'Occam's razor' which states that unnecessary complex models should not be prefered to simpler ones (MacKay 1992). Table 3 shows that the spread of the predictive distribution is increased for non-training inputs in the case of models with more than two hidden units. Therefore Occam's razor is supported by the bootstrap predictive distribution *without* knowing the correct prediction.

This effect shows that bootstrap procedures may be utilized for *model selection*. Analoguous to Liu (1993) we may use a crossvalidation strategy to determine the prediction error for the bootstrap estimate $\hat{\beta}_b$ for sample elements of $X(n)$ which are not contained in the bootstrap sample $X_b^*(n)$. In a similar way Efron (1982, p.52f) determines the error for the predictions $g_{\hat{\beta}_b}(y)$ within the full sample $X(n)$ and uses this as an indicator of the model performance.

# 4   SUMMARY

The bootstrap method offers an computation intensive alternative to estimate the predictive distribution for a neural network even if the analytic derivation is intractable. The available asymptotic results show that it is valid for a large number of linear, nonlinear and even nonparametric regression problems. It has the potential to model the distribution of estimators to a higher precision than the usual normal asymptotics. It even may be valid if the normal asymptotics fail. However, the theoretical properties of bootstrap procedures for neural networks – especially nonlinear models – have to be investigated more comprehensively. In contrast to the Bayesian approach no distributional assumptions (e.g. normal errors) are have to be specified. The simulation experiments show that bootstrap methods offer practical advantages as the performance of the model with respect to a new input may be readily assessed.

### Acknowledgements

This research was supported in part by the German Federal Department of Reserach and Technology, grant ITW8900A7.

# References

Beran, R. (1988): Prepivoting Test Statistics: A Bootstrap View of Asymptotic Refinements. *Journal of the American Statistical Association.* vol. 83, pp.687-697.

Beran, R. (1990): Calibrating Prediction Regions. *Journal of the American Statistical Association.*, vol. 85, pp.715-723.

Bickel, P.J., Freedman, D.H. (1981): Some Asymptotic Theory for the Bootstrap. *The Annals of Statistics*, vol. 9, pp.1196-1217.

Bickel, P.J., Freedman, D.H. (1983): Bootstrapping Regression Models with many Parameters. In P. Bickel, K. Doksum, J.C. Hodges (eds.) *A Festschrift for Erich Lehmann.* Wadsworth, Belmont, CA, pp.28-48.

DiCiccio, T.J., Romano, J.P. (1988): A Review of Bootstrap Confidence Intervals. *J. Royal Statistical Soc.*, Ser. B, vol. 50, pp.338-354.

Efron, B. (1979): Bootstrap Methods: Another Look at the Jackknife. *The Annals of Statistics*, vol 7, pp.1-26.

Efron, B. (1982): *The Jackknife, the Bootstrap and Other Resampling Plans.* SIAM, Philadelphia.

Efron, B., Gong, G. (1983): A leisure look at the bootstrap, the jackknife and crossvalidation. *American Statistician*, vol. 37, pp.36-48.

Efron, B., Tibshirani (1986): Bootstrap methods for Standard Errors, Confidence Intervals, and other Measures of Statistical Accuracy . *Statistical Science*, vol 1, pp.54-77.

Freedman, D.H. (1981): Bootstrapping Regression Models. *The Annals of Statistics*, vol 9, p.1218-1228.

Härdle, W.(1990): *Applied Nonparametric Regression.* Cambridge University Press, Cambridge.

Härdle, W., Mammen, E. (1990): Bootstrap Methods in Nonparametric Regression. *Preprint Nr. 593.* Sonderforschungsbereich 123, University of Heidelberg.

Hall, P. (1988): Theoretical Comparison of Bootstrap Confidence Intervals. *The Annals of Statistics*, vol 16, pp.927-985.

Hinkley, D. . (1988): Bootstrap Methods. *Journal of the Royal Statistical Society*, Ser. B, vol.50, pp.321-337.

Liu, R. (1988): Bootstrap Procedures under some non i.i.d. Models. *The Annals of Statistics*, vol.16, pp. 1696-1708.

Liu, Y. (1993): Neural Network Model Selection Using Asymptotic Jackknife Estimator and Cross-Validation Method. This volume.

MacKay, D. J. C. (1992): Bayesian Model Comparison and Backprop Nets. In Moody, J.E., Hanson, S.J., Lippman, R.P. (eds.) *Advances in Neural Information Processing Systems 4.* Morgan Kaufmann, San Mateo, pp.839-846.

Mammen, E. (1991): When does Bootstrap Work: Asymptotic Results and Simulations. *Preprint Nr. 623.* Sonderforschungsbereich 123, University of Heidelberg.

Pearlmutter, B.A., Rosenfeld, R. (1991): Chaitin-Kolmogorov Complexity and Generalization in Neural Networks. in Lippmann et al. (eds.): *Advances in Neural Information Processing Systems 3*, Morgan Kaufmann, pp.925-931.

C.F.J. Wu (1986): Jackknife, Bootstrap and other Resampling Methods in Regression Analysis. *The Annals of Statistics*, vol. 14, p.1261-1295.
